# From Data Distributions to Regularization in Invariant Learning

Todd K. Leen

Department of Computer Science and Engineering
Oregon Graduate Institute of Science and Technology
20000 N.W. Walker Rd
Beaverton, Oregon 97006
*tleen@cse.ogi.edu*

## Abstract

Ideally pattern recognition machines provide constant output when the inputs are transformed under a group $\mathcal{G}$ of desired invariances. These invariances can be achieved by enhancing the training data to include examples of inputs transformed by elements of $\mathcal{G}$, while leaving the corresponding targets unchanged. Alternatively the cost function for training can include a regularization term that penalizes changes in the output when the input is transformed under the group.

This paper relates the two approaches, showing precisely the sense in which the regularized cost function approximates the result of adding transformed (or distorted) examples to the training data. The cost function for the enhanced training set is equivalent to the sum of the original cost function plus a regularizer. For unbiased models, the regularizer reduces to the intuitively obvious choice – a term that penalizes changes in the output when the inputs are transformed under the group. For infinitesimal transformations, the coefficient of the regularization term reduces to the variance of the distortions introduced into the training data. This correspondence provides a simple bridge between the two approaches.

# 1   Approaches to Invariant Learning

In machine learning one sometimes wants to incorporate invariances into the function learned. Our knowledge of the problem dictates that the machine outputs ought to remain constant when its inputs are transformed under a set of operations $\mathcal{G}$[1]. In character recognition, for example, we want the outputs to be invariant under shifts and small rotations of the input image.

In neural networks, there are several ways to achieve this invariance

1. The invariance can be hard-wired by weight sharing in the case of summation nodes (LeCun et al. 1990) or by constraints similar to weight sharing in higher-order nodes (Giles et al. 1988).

2. One can enhance the training ensemble by adding examples of inputs transformed under the desired invariance group, while maintaining the same targets as for the raw data.

3. One can add to the cost function a regularizer that penalizes changes in the output when the input is transformed by elements of the group (Simard et al. 1992).

Intuitively one expects the approaches in 3 and 4 to be intimately linked. This paper develops that correspondence in detail.

# 2   The Distortion-Enhanced Input Ensemble

Let the input data $x$ be distributed according to the density function $p(x)$. The conditional distribution for the corresponding targets is denoted $p(t|x)$. For simplicity of notation we take $t \in R$. The extension to vector targets is trivial. Let $f(x; w)$ denote the network function, parameterized by weights $w$. The training procedure is assumed to minimize the expected squared error

$$\mathcal{E}(w) \equiv \int \int dt\, dx\, p(t|x)\, p(x)\, [\, t - f(x; w)\, ]^2 \quad . \tag{1}$$

We wish to consider the effects of adding new inputs that are related to the old by transformations that correspond to the desired invariances. These transformations, or distortions, of the inputs are carried out by group elements $g \in \mathcal{G}$. For Lie groups[2], the transformations are analytic functions of parameters $\alpha \in R^k$

$$x \to x' = g(x; \alpha) \quad , \tag{2}$$

with the identity transformation corresponding to parameter value zero

$$g(x; 0) = x \quad . \tag{3}$$

In image processing, for example, we may want our machine to exhibit invariance with respect to rotation, scaling, shearing and translations of the plane . These

transformations form a six-parameter Lie group[3].

By adding distorted input examples we alter the original density $p(x)$. To describe the new density, we introduce a probability density for the transformation parameters $p(\alpha)$. Using this density, the distribution for the distortion-enhanced input ensemble is

$$
\begin{aligned}
p(x') &= \int \int d\alpha \, dx \, p(x'|x,\alpha) \, p(\alpha) \, p(x) \\
&= \int \int d\alpha \, dx \, \delta(\, x' - g(x;\alpha)\,) \, p(\alpha) \, p(x) \ ,
\end{aligned}
$$

where $\delta(\cdot)$ is the Dirac delta function[4]

Finally we impose that the targets remain unchanged when the inputs are transformed according to (2) *i.e.*, $p(t|x') = p(t|x)$. Substituting $p(x')$ into (1) and using the invariance of the targets yields the cost function

$$
\tilde{\mathcal{E}} = \int \int \int dt \, dx \, d\alpha \ p(t|x) \, p(x) \, p(\alpha) \ [\, t - f(g(x;\alpha);w)\,]^2 \ . \tag{4}
$$

Equation (4) gives the cost function for the distortion-enhanced input ensemble.

## 3 Regularization and Hints

The remainder of the paper makes precise the connection between adding transformed inputs, as embodied in (4), and various regularization procedures. It is straightforward to show that the cost function for the distortion-enhanced ensemble is equivalent to the cost function for the original data ensemble (1) plus a regularization term. Adding and subtracting $f(x;w)$ to the term in square brackets in (4), and expanding the quadratic leaves

$$
\tilde{\mathcal{E}} = \mathcal{E} + \mathcal{E}_R \ , \tag{5}
$$

where the regularizer is

$$
\begin{aligned}
\mathcal{E}_R &= \mathcal{E}_H + \mathcal{E}_C \\
&\equiv \int d\alpha \, p(\alpha) \int dx \ p(x) \ [\, f(x,w) - f(g(x;\alpha);w)\,]^2 \\
&\quad - 2 \int \int \int dt \, dx \, d\alpha \ p(t|x) \, p(x) \, p(\alpha) \\
&\qquad \times [\, t - f(x;w)\,]\,[\, f(g(x;\alpha);w) - f(x;w)\,] \ . \tag{6}
\end{aligned}
$$

Training with the original data ensemble using the cost function (5) is equivalent to adding transformed inputs to the data ensemble.

The first term of the regularizer $\mathcal{E}_H$ penalizes the average squared difference between $f(x; w)$ and $f(g(x; \alpha); w)$. This is *exactly* the form one would intuitively apply in order to insure that the network output not change under the transformation $x \rightarrow g(x, \alpha)$. Indeed this is the similar to the form of the invariance "hint" proposed by Abu-Mostafa (1993). The difference here is that there is no arbitrary parameter multiplying the term. Instead the strength of the regularizer is governed by the average over the density $p(\alpha)$. The term $\mathcal{E}_H$ measures the error in satisfying the invariance hint.

The second term $\mathcal{E}_C$ measures the correlation between the error in fitting to the data, and the error in satisfying the hint. Only when these correlations vanish is the cost function for the enhanced ensemble equal to the original cost function plus the invariance hint penalty.

The correlation term vanishes trivially when either

1. The invariance $f(g(x; \alpha); w) = f(x; w)$ is satisfied, or

2. The network function equals the least squares regression on $t$

$$f(x; w) \; = \; \int dt \, p(t|x) \, t \; \equiv \; E[t|x] \; . \tag{7}$$

The lowest possible $\mathcal{E}$ occurs when $f$ satisfies (7), at which $\mathcal{E}$ becomes the variance in the targets averaged over $p(x)$. By substituting this into $\mathcal{E}_C$ and carrying out the integration over $dt \, p(t|x)$, the correlation term is seen to vanish.

If the minimum of $\tilde{\mathcal{E}}$ occurs at a weight for which the invariance is satisfied (condition 1 above), then minimizing $\tilde{\mathcal{E}}(w)$ is equivalent to minimizing $\mathcal{E}(w)$. If the minimum of $\tilde{\mathcal{E}}$ occurs at a weight for which the network function is the regression (condition 2), then minimizing $\tilde{\mathcal{E}}$ is equivalent to minimizing the cost function with the intuitive regularizer $\mathcal{E}_H$ [5].

## 3.1  Infinitesimal Transformations

Above we enumerated the conditions under which the correlation term in $\mathcal{E}_R$ vanishes exactly for unrestricted transformations. If the transformations are analytic in the parameters $\alpha$, then by restricting ourselves to small transformations (those close to the identity) we can show how the correlation term approximately vanishes for unbiased models. To implement this, we assume that $p(\alpha)$ is sharply peaked up about the origin so that large transformations are unlikely.

We obtain an approximation to the cost function $\tilde{\mathcal{E}}$ by expanding the integrands in (6) in power series about $\alpha = 0$ and retaining terms to second order. This leaves

$$
\tilde{\mathcal{E}} = \mathcal{E} + \int \int dx\, d\alpha\, p(x)\, p(\alpha) \left( \alpha_i \left. \frac{\partial g^\mu}{\partial \alpha_i} \right|_{\alpha=0} \frac{\partial f}{\partial x^\mu} \right)^2
$$

$$
- 2 \int \int \int dt\, dx\, d\alpha\; p(t|x)\, p(x)\, p(\alpha)\, [t - f(x; w)] \times
$$

$$
\left[ \left( \alpha_i \left. \frac{\partial g^\mu}{\partial \alpha_i} \right|_{\alpha=0} + \frac{1}{2} \alpha_i \alpha_j \left. \frac{\partial^2 g^\mu}{\partial \alpha_i\, \partial \alpha_j} \right|_{\alpha=0} \right) \left( \frac{\partial f}{\partial x^\mu} \right) \right.
$$

$$
\left. + \frac{1}{2} \alpha_i \alpha_j \left. \frac{\partial g^\mu}{\partial \alpha_i} \right|_{\alpha=0} \left. \frac{\partial g^\nu}{\partial \alpha_j} \right|_{\alpha=0} \left( \frac{\partial^2 f}{\partial x^\nu\, \partial x^\mu} \right) \right] + \mathcal{O}(\alpha^3) \qquad (8)
$$

where $x^\mu$ and $g^\mu$ denote the $\mu^{th}$ components of $x$ and $g$, $\alpha_i$ denotes the $i^{th}$ component of the transformation parameter vector, repeated Greek and Roman indices are summed over, and all derivatives are evaluated at $\alpha = 0$. Note that we have used the fact that Lie group transformations are analytic in the parameter vector $\alpha$ to derive the expansion.

Finally we introduce two assumptions on the distribution $p(\alpha)$. First $\alpha$ is assumed to be zero mean. This corresponds, in the linear approximation, to a distribution of distortions whose mean is the identity transformation. Second, we assume that the components of $\alpha$ are uncorrelated so that the covariance matrix is diagonal with elements $\sigma_i^2$, $i = 1 \ldots k$.[6] With these assumptions, the cost function for the distortion-enhanced ensemble simplifies to

$$
\tilde{\mathcal{E}} = \mathcal{E} + \sum_{i=1}^{k} \sigma_i^2 \int dx\, p(x) \left( \left. \frac{\partial g^\mu}{\partial \alpha_i} \right|_{\alpha=0} \frac{\partial f}{\partial x^\mu} \right)^2
$$

$$
- \sum_{i=1}^{k} \sigma_i^2 \int \int dx\, dt\; p(t|x)\, p(x) \left\{ (f(x; w) - t) \right.
$$

$$
\times \left[ \left. \frac{\partial^2 g^\mu}{\partial \alpha_i^2} \right|_{\alpha=0} \left( \frac{\partial f}{\partial x^\mu} \right) + \left. \frac{\partial g^\mu}{\partial \alpha_i} \right|_{\alpha=0} \left. \frac{\partial g^\nu}{\partial \alpha_i} \right|_{\alpha=0} \left( \frac{\partial^2 f}{\partial x^\nu\, \partial x^\mu} \right) \right] \right\}
$$

$$
+ \mathcal{O}(\sigma^4) \; . \qquad (9)
$$

This last expression provides a simple bridge between the the methods of adding transformed examples to the data, and the alternative of adding a regularizer to the cost function: The coefficient of the regularization term in the latter approach is equal to the *variance of the transformation parameters* in the former approach.

### 3.1.1  Unbiased Models

For unbiased models the regularizer in $\tilde{\mathcal{E}}(w)$ assumes a particularly simple form. Suppose the network function is rich enough to form an unbiased estimate of the least squares regression on $t$ for the *undistorted* data ensemble. That is, there exists a weight value $w_0$ such that

$$f(x; w_0) = \int dt \; t \; p(t|x) \equiv E[t \,|\, x] \; . \tag{10}$$

This is the *global minimum* for the original error $\mathcal{E}(w)$.

The arguments of section 3 apply here as well. However we can go further. Even if there is only a single, isolated weight value for which (10) is satisfied, then to $\mathcal{O}(\sigma^2)$ the correlation term in the regularizer vanishes. To see this note that by the implicit function theorem the modified cost function (9) has its global minimum at the new weight[7]

$$\tilde{w}_0 = w_0 + \mathcal{O}(\sigma^2) \; . \tag{11}$$

At this weight, the network function is no longer the regression on $t$, but rather

$$f(x; \tilde{w}_0) = E[t \,|\, x] + \mathcal{O}(\sigma^2) \; . \tag{12}$$

Substituting (12) into (9), we find that the minimum of (9) is, to $\mathcal{O}(\sigma^2)$, at the same weight as the minimum of

$$\hat{\mathcal{E}} = \mathcal{E} + \sum_{i=1}^{k} \sigma_i^2 \int dx \, p(x) \left[ \left. \frac{\partial g^\mu}{\partial \, \alpha_i} \right|_{\alpha=0} \frac{\partial f(x,w)}{\partial x^\mu} \right]^2 \; . \tag{13}$$

To $\mathcal{O}(\sigma^2)$, minimizing (13) is equivalent to minimizing (9). So we regard $\hat{\mathcal{E}}$ as the effective cost function.

The regularization term in (13) is proportional to the average square of the gradient of the network function along the direction in the input space generated by the linear part of $g$. The quantity inside the square brackets is just the linear part of $[f(g(x; \alpha)) - f(x)]$ from (6). The magnitude of the regularization term is just the variance of the distribution of distortion parameters.

This is precisely the form of the regularizer given by Simard et al. in their tangent prop algorithm (Simard et al, 1992). This derivation shows the equivalence (to $\mathcal{O}(\sigma^2)$) between the tangent prop regularizer and the alternative of modifying the input distribution. Furthermore, we see that with this equivalence, the constant fixing the strength of the regularization term is simply the *variance of the distortions* introduced into the original training set.

We should stress that the equivalence between the regularizer, and the distortion-enhanced ensemble in (13) only holds to $\mathcal{O}(\sigma^2)$. If one allows the variance of the

distortion parameters $\sigma^2$ to become arbitrarily large in an effort to mock up an arbitrarily large regularization term, then the equivalence expressed in (13) breaks down since terms of order $\mathcal{O}(\sigma^4)$ can no longer be neglected. In addition, if the transformations are to be kept small so that the linearization holds (e.g. by restricting the density on $\alpha$ to have support on a small neighborhood of zero), then the variance will bounded above.

### 3.1.2 Smoothing Regularizers

In the previous sections we showed the equivalence between modifying the input distribution and adding a regularizer to the cost function. We derived this equivalence to illuminate mechanisms for obtaining invariant pattern recognition. The technique for dealing with infinitesimal transformations in section §3.1 was used by Bishop (1994) to show the equivalence between added input noise and smoothing regularizers. Bishop's results, though they preceded our own, are a special case of the results presented here. Suppose the group $\mathcal{G}$ is restricted to translations by random vectors $g(x; \alpha) = x + \alpha$ where $\alpha$ is spherically distributed with variance $\sigma_\alpha^2$. Then the regularizer in (13) is

$$\mathcal{E}_R = \sigma_\alpha^2 \int dx \; p(x) \; |\nabla_x f(x; w)|^2 \; . \qquad (14)$$

This regularizer penalizes large magnitude gradients in the network function and is, as pointed out by Bishop, one of the class of generalized Tikhonov regularizers.

## 4  Summary

We have shown that enhancing the input ensemble by adding examples transformed under a group $x \to g(x; \alpha)$, while maintaining the target values, is equivalent to adding a regularizer to the original cost function. For unbiased models the regularizer reduces to the intuitive form that penalizes the mean squared difference between the network output for transformed and untransformed inputs – i.e. the error in satisfying the desired invariance. In general the regularizer includes a term that measures correlations between the error in fitting the data, and the error in satisfying the desired invariance. For infinitesimal transformations, the regularizer is equivalent (up to terms linear in the variance of the transformation parameters) to the tangent prop form given by Simard et al. (1992), with regularization coefficient equal to the variance of the transformation parameters. In the special case that the group transformations are limited to random translations of the input, the regularizer reduces to a standard smoothing regularizer.

We gave conditions under which enhancing the input ensemble and adding the intuitive regularizer $\mathcal{E}_H$ are equivalent. However this equivalence is only with regard to the optimal weight. We have not compared the training dynamics for the two approaches. In particular, it is quite possible that the full regularizer $\mathcal{E}_H + \mathcal{E}_C$ exhibits different training dynamics from the intuitive form $\mathcal{E}_H$. For the approach in which data are added to the input ensemble, one can easily construct datasets and distributions $p(\alpha)$ that either increase the condition number of the Hessian, or decrease it. Finally, it may be that the intuitive regularizer can have either detrimental or positive effects on the Hessian as well.

**Acknowledgments**

I thank Lodewyk Wessels, Misha Pavel, Eric Wan, Steve Rehfuss, Genevieve Orr and Patrice Simard for stimulating and helpful discussions, and the reviewers for helpful comments. I am grateful to my father for what he gave to me in life, and for the presence of his spirit after his recent passing.

This work was supported by EPRI under grant RP8015-2, AFOSR under grant FF4962-93-1-0253, and ONR under grant N00014-91-J-1482.

**References**

Yasar S. Abu-Mostafa. A method for learning from hints. In S. Hanson, J. Cowan, and C. Giles, editors, *Advances in Neural Information Processing Systems, vol. 5*, pages 73–80. Morgan Kaufmann, 1993.

Chris M. Bishop. Training with noise is equivalent to Tikhonov regularization. To appear in *Neural Computation*, 1994.

C.L. Giles, R.D. Griffin, and T. Maxwell. Encoding geometric invariances in higher-order neural networks. In D.Z.Anderson, editor, *Neural Information Processing Systems*, pages 301–309. American Institute of Physics, 1988.

Y. Le Cun, B. Boser, J.S. Denker, D. Henderson, R.E. Howard, W. Hubbard, and L.D. Jackel. Handwritten digit recognition with a back-propagation network. In *Advances in Neural Information Processing Systems, vol. 2*, pages 396–404. Morgan Kaufmann Publishers, 1990.

Patrice Simard, Bernard Victorri, Yann Le Cun, and John Denker. Tangent prop - a formalism for specifying selected invariances in an adaptive network. In John E. Moody, Steven J. Hanson, and Richard P. Lippmann, editors, *Advances in Neural Information Processing Systems 4*, pages 895–903. Morgan Kaufmann, 1992.

D.H. Sattinger and O.L. Weaver. *Lie Groups and Algebras with Applications to Physics, Geometry and Mechanics*. Springer-Verlag, 1986.

## Footnotes

[1]We assume that the set forms a group.

[2]See for example (Sattinger, 1986).

[3]The parameters for rotations, scaling and shearing completely specify elements of $GL2$, the four parameter group of $2 \times 2$ invertible matrices. The translations carry an additional two degrees of freedom.

[4]In general the density on $\alpha$ might vary through the input space, suggesting the conditional density $p(\alpha\,|\,x)$. This introduces rather minor changes in the discussion that will not be considered here.

[5]If the data is to be fit optimally, with enough freedom left over to satisfy the invariance hint, then there must be several weight values (perhaps a continuum of such values) for which the network function satisfies (7). That is, the problem must be under-specified. If this is the case, then the interesting part weight space is just the subset on which (7) is satisfied. On this subset the correlation term in (6) vanishes and the regularizer assumes the intuitive form.

[6]Note that the *transformed patterns* may be correlated in parts of the pattern space. For example the results of applying the shearing and rotation operations to an infinite vertical line are indistinguishable. In general, there may be regions of the pattern space for which the action of several different group elements are indistinguishable; that is $x' = g(x; \alpha) = g(x; \beta)$. However this does not imply that $\alpha$ and $\beta$ are statistically correlated.

[7]We assume that the Hessian of $\mathcal{E}$ is nonsingular at $w_0$.
